# A comparison between a neural network model for the formation of brain maps and experimental data

**K. Obermayer**
Beckman-Institute
University of Illinois
Urbana, IL 61801

**K. Schulten**
Beckman-Institute
University of Illinois
Urbana, IL 61801

**G.G. Blasdel**
Harvard Medical School
Harvard University
Boston, MA 02115

## Abstract

Recently, high resolution images of the simultaneous representation of orientation preference, orientation selectivity and ocular dominance have been obtained for large areas in monkey striate cortex by optical imaging [1-3]. These data allow for the first time a "local" as well as "global" description of the spatial patterns and provide strong evidence for correlations between orientation selectivity and ocular dominance.

A quantitative analysis reveals that these correlations arise when a five-dimensional feature space (two dimensions for retinotopic space, one each for orientation preference, orientation specificity, and ocular dominance) is mapped into the two available dimensions of cortex while locally preserving topology. These results provide strong evidence for the concept of topology preserving maps which have been suggested as a basic design principle of striate cortex [4-7].

Monkey striate cortex contains a retinotopic map in which are embedded the highly repetitive patterns of orientation selectivity and ocular dominance. The retinotopic projection establishes a "global" order, while maps of variables describing other stimulus features, in particular line orientation and ocularity, dominate cortical organization locally. A large number of pattern models [8-12] as well as models of development [6,7,13-21] have been proposed to describe the spatial structure of these patterns and their development during ontogenesis. However, most models have not been compared with experimental data in detail. There are two reasons for this: (*i*) many model-studies were not elaborated enough to be experimentally testable and (*ii*) a sufficient amount of experimental data obtained from large areas of striate cortex was not available.

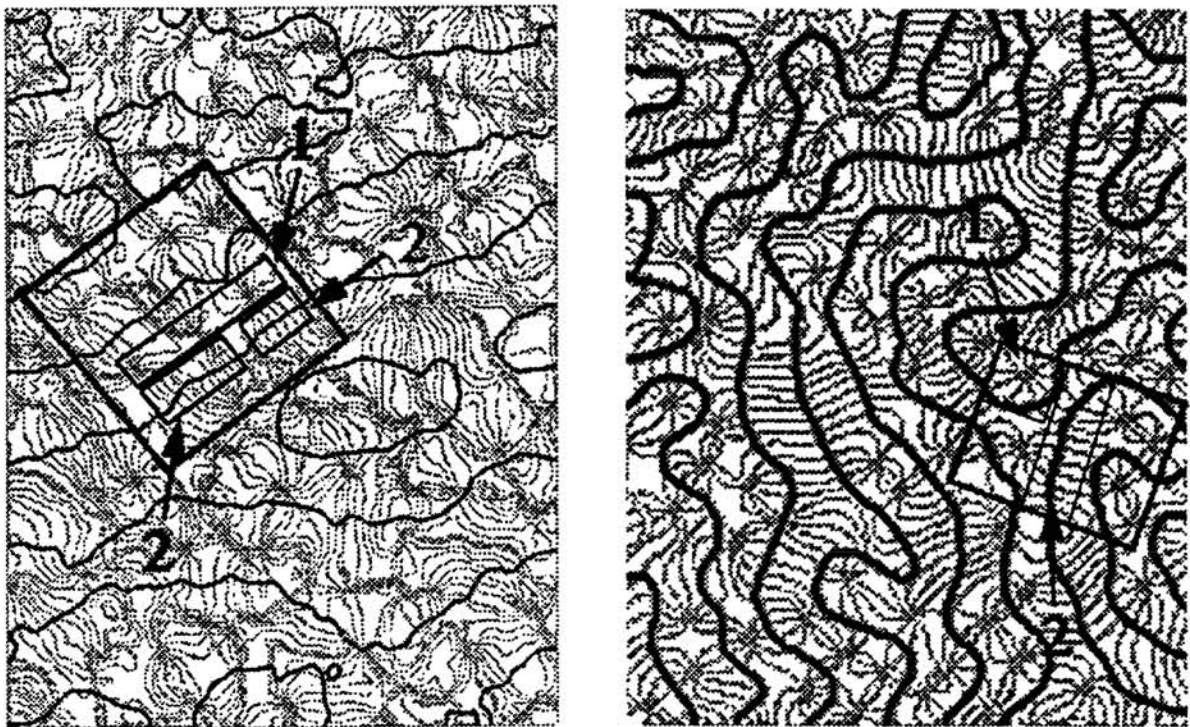

**Figure 1:** Spatial pattern of orientation preference and ocular dominance in monkey striate cortex (left) compared with predictions of the SOFM-model (right). Iso-orientation lines (gray) are drawn in intervals of $11.25°$ (left) and $18.0°$ (right), respectively. Black lines indicate the borders $(w_5(\vec{r}) = 0)$ of ocular dominance bands. The areas enclosed by black rectangles mark corresponding elements of organization in monkey striate cortex and in the simulation result (see text). **Left:** Data obtained from a 3.1mm × 4.2mm patch of the striate cortex of an adult macaque (macaca nemestrina) by optical imaging [1-3]. The region is located near the border with area 18, close to midline. **Right:** Model-map generated by the SOFM-algorithm. The figure displays a small section of a network of size $N = d = 512$. The parameters of the simulation were: $\varepsilon = 0.02$, $\sigma_h = 5$, $v_{3,4}^{max} = 20.48$, $v_5^{max} = 15.36$, $9 \cdot 10^7$ iterations, with retinotopic initial conditions and periodic boundary conditions.

# 1    Orientation and ocular dominance columns in monkey striate cortex

Recent advances in optical imaging [1-3,22,23] now make it possible to obtain high resolution images of the spatial pattern of orientation selectivity and ocular dominance from large cortical areas. Prima vista analysis of data from monkey striate cortex reveals that the spatial pattern of orientation preference and ocular dominance is continuous and highly repetitive across cortex. On a global scale orientation preferences repeat along every direction of cortex with similar periods. Locally, orientation preferences are organized as parallel *slabs* (arrow 1, Fig. 1a) in *linear zones*, which start and end at *singularities* (arrow 2, Fig. 1a), point-like discontinuities, around which orientation preferences change by $\pm180°$ in a pinwheel-like fashion. Both types of singularities appear in equal numbers (359:354 for maps obtained from four adult macaques) with a density of $5.5/mm^2$ (for regions close to

| Fourier transforms | $w_j(\vec{k})$ | $= \sum_{\vec{r}} \exp(i\vec{k}\vec{r})\, w_j(\vec{r})$ |
|---|---|---|
| correlation functions | $C_{ij}(\vec{p})$ | $= <w_i(\vec{r})\, w_j(\vec{r}+\vec{p})>_{\vec{r}}$ |
| feature gradients | $|\nabla_{\vec{r}} w_j(\vec{r})|$ | $= \{(w_j(r_1+1,r_2) - w_j(r_1,r_2))^2$ $+ (w_j(r_1,r_2+1) - w_j(r_1,r_2))^2\}^{1/2}$ |
| Gabor transforms | $g_j(\vec{k},\vec{r})$ | $= (2\pi\sigma_g^2)^{-\frac{1}{4}} \int d^2r'\, w_j(\vec{r}')$ $\exp\{-\frac{(\vec{r}-\vec{r}')^2}{4\sigma_g^2} + i\vec{k}(\vec{r}' - \frac{1}{2}\vec{r})\}$ |

**Table 1:** Quantitative measures used to characterize cortical maps.

the midline). Figure 1a reveals that the iso-orientation lines cross ocular dominance bands at nearly right angles most of the time (region number 2) and that singularities tend to align with the centers of the ocular dominance bands (region number 1). Where orientation preferences are organized as parallel slabs (region number 2), the iso-orientation contours are often equally spaced and orientation preferences change linearly with distance.

These results are confirmed by a quantitative analysis (see Table 1). For the following we denote cortical location by a two-dimensional vector $\vec{r}$. At each location we denote the (average) position of receptive field centroids in visual space by $(w_1(\vec{r}), w_2(\vec{r}))$. Orientation selectivity is described by a two-dimensional vector $(w_3(\vec{r}), w_4(\vec{r}))$, whose length and direction code for orientation tuning strength and preferred orientation, respectively [1,10]. Ocular dominance is described by a real-valued function $w_5(\vec{r})$, which denotes the difference in response to stimuli presented to the left and right eye. Data acquisition and postprocessing are described in detail in [1-3].

A Fourier transform of the map of orientation preferences reveals a spectrum which is a nearly circular band (Fig. 2a), showing that orientation preferences repeat with similar periods in every direction in cortex. Neglecting the slight anisotropy in the experimental data[1], a power spectrum can be approximated by averaging amplitudes over all directions of the wave-vector (Fig. 2b, dots). The location of the peak corresponds to an average period $\lambda_0 = 710\mu m \pm 50\mu m$[2] and it's width to a *coherence length* of $820\mu m \pm 130\mu m$. The coherence length indicates the typical distance over which orientation preferences can change linearly and corresponds to the average size of linear zones in Fig. 1a. The corresponding autocorrelation functions (Fig. 2c) have a Mexican hat shape. The minimum occurs near $300\mu m$, which indicates that orientation preferences in regions separated by this distance tend to be orthogonal. In summary, the spatial pattern of orientation preference is characterized by local correlation and global "disorder".

**a)**

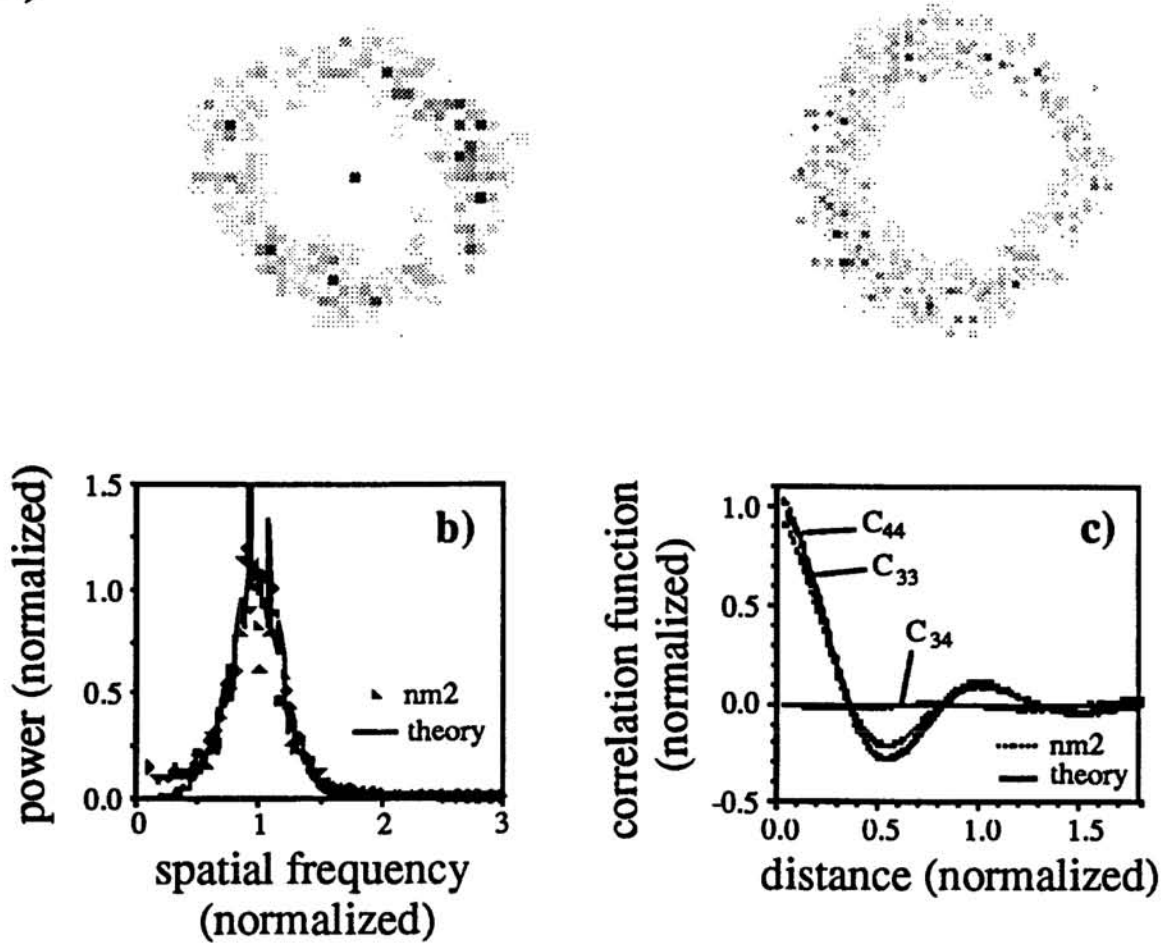

**Figure 2**: Fourier analysis and correlation functions of the orientation map in monkey striate cortex (animal nm2) compared with the predictions of the SOFM-model. Simulation results were taken from the data set described in Fig. 1, right. **(a)** Fourier spectra of nm2 **(left)** and simulation results **(right)**. Each pixel represents one mode; location and gray value of the pixel indicate wave-vector and energy, respectively. **(b)** Approximate power spectrum (normalized) obtained by averaging the Fourier-spectra in (a) over all directions of the wave-vector. Peak frequency of 1.0 corresponds to 1.4/mm for nm2. **(c)** Correlation functions (normalized). A distance of 1.0 corresponds to $725\mu$m for nm2.

Local properties of the spatial patterns, as well as correlations between orientation preference and ocular dominance, can be quantitatively characterized using Gabor-Helstrom-transforms (see Table 1). If the radius $\sigma_g$ of the Gaussian function in the Gabor-filter is smaller than the coherence length the Gabor-transform of any of the quantities $w_3(\vec{r})$, $w_4(\vec{r})$ and $w_5(\vec{r})$ typically consists of two localized regions of high energy located on opposite sides of the origin. The length $|\vec{k}_i|$ of the vectors $\vec{k}_i$, $i \in [3, 4, 5]$, which corresponds to the centroids of these regions, fluctuates around the characteristic wave-number $2\pi/\lambda_0$ of this pattern, and its direction gives the normal to the ocular dominance bands and iso-orientation slabs at the location $\vec{r}$, where the Gabor-transform was performed.

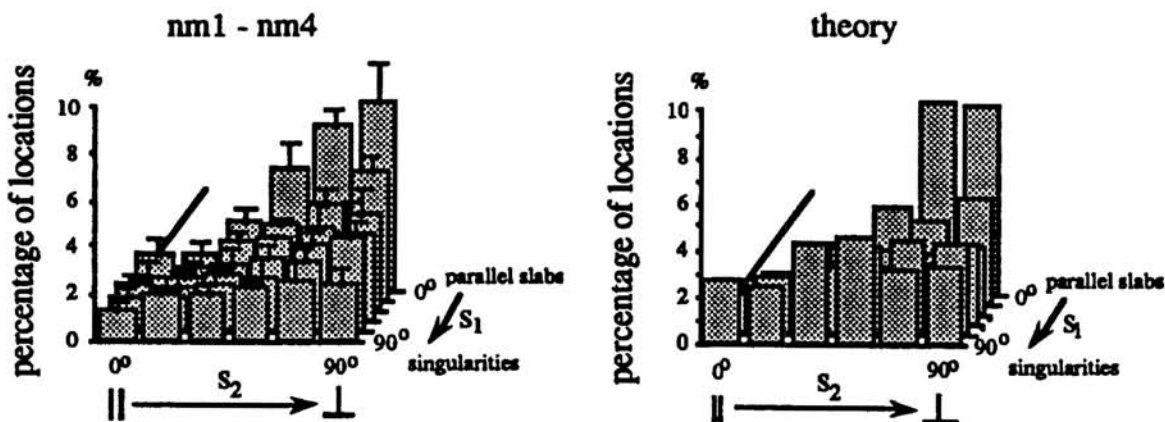

**Figure 3**: Gabor-analysis of cortical maps. The percentage of map locations is plotted against the parameters $s_1$ and $s_2$ (see text) for 3,421 locations randomly selected from the cortical maps of four monkeys, nm1-nm4, (**left**) and for 1,755 locations randomly selected from simulation results (**right**). Error bars indicate standard deviations. Simulation results were taken from the data set described in Fig. 1. $\sigma_g$ was $150\mu m$ for the experimental data and 28 pixels for the SOFM-map.

Results of this analysis are shown in Fig. 3 (left) for 3,434 samples selected randomly from data of four animals. The angle between $\vec{k}_3$ and $\vec{k}_4$ is represented along the $s_1$ axis. Histograms at the back, where $s_1 = 0°$, represent regions where iso-orientation lines are parallel. Histograms in the front, where $s_1 = 90°$, represent regions containing singularities. The intersection angle of iso-orientation slabs and ocular dominance bands is represented along the $s_2$ axis. The proportion of sampled regions increases steadily with decreasing $s_1$. As $s_1$ approaches zero, values accumulate at the right, where orientation and ocular dominance bands are orthogonal. Thus linear zones *and* singularities are important elements of cortical organization but linear zones (back rows) are the most prominent features in monkey striate cortex[3]. Where iso-orientation regions are organized as parallel slabs, orientation slabs intersect ocular dominance bands at nearly right angles (back and right corner of diagrams).

## 2   Topology preserving maps

Recently, topology preserving maps have been suggested as a basic design principle underlying these patterns and its was proposed that these maps are generated by simple and biologically plausible pattern formation processes [4,6,7]. In the following we will test these models against the recent experimental data.

We consider a five-dimensional *feature space* $V$ which is spanned by quantities describing the most prominent receptive field properties of cortical cells: position of a receptive field in retinotopic space $(v_1, v_2)$, orientation preference and tuning strength $(v_3, v_4)$, and ocular dominance $(v_5)$. If all combinations of these properties

are represented in striate cortex, each point in this five-dimensional feature space is mapped onto one point on the two-dimensional *cortical surface A*.

In order to generate these maps we employ the feature map (SOFM-) algorithm of Kohonen [15,16] which is known to generate topology preserving maps between spaces of different dimensionality [4,5][4]. The algorithm describes the development of these patterns as unsupervised learning, i.e. the features of the input patterns determine the features to be represented in the network [4]. Mathematically, the algorithm assignes *feature vectors* $\vec{w}(\vec{r})$, which are points in the feature space, to cortical *units* $\vec{r}$, which are points on the cortical surface. In our model the surface is divided into $N \times N$ small patches, *units* $\vec{r}$, which are arranged on a two-dimensional lattice (network layer) with periodic boundary conditions (to avoid edge effects). The average receptive field properties of neurons located in each patch are characterized by the feature vector $\vec{w}(\vec{r})$ whose components $(\vec{w}_j(\vec{r})$ are interpreted as receptive field properties of these neurons. The algorithm follows an iterative procedure. At each step an *input vector* $\vec{v}$, which is of the same dimensionality as $\vec{w}(\vec{r})$ is chosen at random according to a probability distribution $P(\vec{v})$. Then the unit $\vec{s}$ whose feature vector $\vec{w}(\vec{s})$ is closest to the input pattern $\vec{v}$ is selected and the components $(\vec{w}_j(\vec{r})$ of its feature vector are changed according to the feature map learning rule [15,16],

$$\Delta \vec{w}(\vec{r}) = \varepsilon \exp\left(-(r_1 - s_1)^2/\sigma_{h1}^2 - (r_2 - s_2)^2/\sigma_{h2}^2\right)(\vec{v} - \vec{w}(\vec{r})), \qquad (1)$$

$P(\vec{v})$ was chosen to be constant within a cylindrical manifold in feature space,

$$V = \{\vec{v} \mid w_1(\vec{r}), w_2(\vec{r}) \in [0, d]; \; |(w_3(\vec{r}), w_4(\vec{r}))| < v_{3,4}^{max}; \; |w_5(\vec{r})| < v_5^{max}\}, \qquad (2)$$

where $v_{3,4}^{max}$ and $v_5^{max}$ are some real constants, and zero elsewhere.

Figure 4 shows a typical map, a surface in feature space, generated by the SOFM-algorithm. For the sake of illustration the five-dimensional feature space is projected onto a three-dimensional subspace spanned by the coordinate-axes corresponding to retinotopic location ($v_1$ and $v_2$) and ocular dominance ($v_5$). The locations of feature vectors assigned to the cortical units are indicated by the intersections of a grid in feature space. Preservation of topology requires that the feature vectors assigned to neighboring cortical units must locally have equal distance and must be arranged on a planar square lattice in feature space. Consequently, large changes in one feature, e.g. ocular dominance $v_5$, along a given direction on the network correlate with small changes of the other features, e.g. retinotopic location $v_1$ and $v_2$, along the same direction (crests and troughs of the waves in Fig. 4) and vice versa. Other correlations arise at points where the map exhibits maximal changes in two features. For example for retinotopic location ($v_1$) and ocular dominance ($v_5$) to vary at a maximal rate, the surface in Fig. 4 must be parallel to the ($v_1, v_5$)-plane. Obviously, at such points the directions of maximal change of retinotopic location and ocular dominance are orthogonal on the surface.

In order to compare model predictions with experimental data the surface in the five-dimensional feature space has to be projected into the three-dimensional subspace

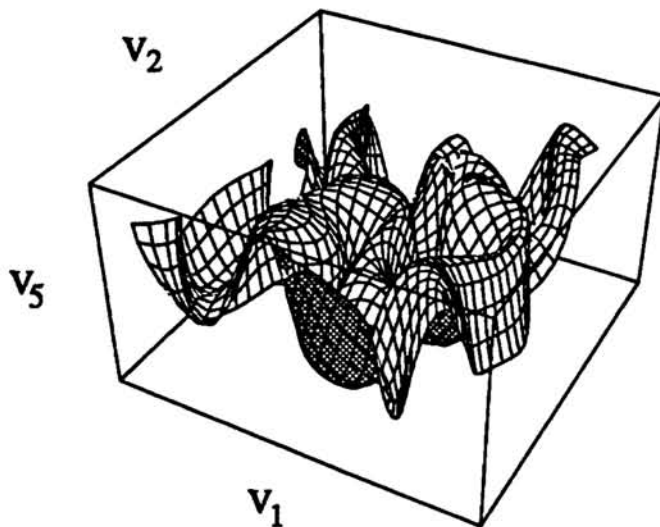

**Figure 4:** Typical map generated by the SOFM-algorithm. The five-dimensional feature space is projected into the three-dimensional subspace spanned by the three coordinates ($v_1$, $v_2$ and $v_5$). Locations of feature vectors which are mapped to the units in the network are indicated by the intersections of a grid in feature space. Only every fourth vector is shown.

spanned by orientation preferences ($v_3$ and $v_4$) and ocular dominance ($v_5$). This projection cannot be visualized easily because the surface completely fills space, intersecting itself multiple times. However, the same line of reasoning applies: (*i*) regions where orientation preferences change quickly, correlate with regions where ocular dominance changes slowly, and (*ii*) in regions where orientation preferences change most rapidly along one direction, ocular dominance has to change most rapidly along the orthogonal direction. Consequently we expect discontinuities of the orientation map to be located in the centers of the ocular dominance bands and iso-orientation slabs to intersect ocular dominance bands at steep angles.

Figures 1, 2 and 3 show simulation results in comparison with experimental data. The algorithm generates all the prominent features of lateral cortical organization: singularities (arrow 1), linear zones (arrow 2), and parallel ocular dominance bands. Singularities are aligned with the centers of ocular dominance bands (region 1) and iso-orientation slabs intersect ocular dominance stripes at nearly right angles (region 2). The shape of Fourier- and power-spectra as well as of the correlation functions agrees quantitatively with the experimental data (see Fig. 2). Isotropic spectra are the result of the invariance of eqs. (1) and (2) under rotation with respect to cortical coordinates $\vec{r}$; global disorder and singularities are a consequence of their invariance under translation. The emergence of singularities can also be understood from an entropy argument. Since dimension reducing maps, which exhibit these features, have increased entropy, they are generated with higher probability. Correlations between orientation preference and ocular dominance, however, follow from geometrical constraints and are inherent properties the topology preserving maps.

## 3   Conclusions

On the basis of our findings the following picture of orientation and ocular dominance columns in monkey striate cortex emerges. Orientation preferences are organized into linear zones *and* singularities, but areas where iso-orientation regions form parallel slabs are apparent across most of the cortical surface. In linear zones,

iso-orientation slabs indeed intersect ocular dominance slabs at right angles as initially suggested by Hubel and Wiesel [8]. Orientation preferences, however, are arranged in an orderly fashion only in regions 0.8mm in size, and the pattern is characterized by local correlation and global disorder.

These patterns can be explained as the result of topology-preserving, dimension reducing maps. Local correlations follow from geometrical constraints and are a direct consequence of the principle of dimension reduction. Global disorder and singularities are consistent with this principle but reflect their generation by a local and stochastic self-organizing process.

## Acknowledgements

The authors would like to thank H. Ritter for fruitful discussions and comments and the Boehringer-Ingelheim Fonds for financial support by a scholarship to K. O. This research has been supported by the National Science Foundation (grant numbers DIR 90-17051 and DIR 91-22522). Computer time on the Connection Machine CM-2 has been made available by the National Center for Supercomputer Applications at Urbana-Champaign funded by NSF.

## Footnotes

[1] Along axes parallel to the ocular dominance slabs, orientation preferences repeat on average every $660\mu m \pm 40\mu m$; perpendicular to the stripes every $840\mu m \pm 40\mu m$. The slight horizontal elongation reflects the fact that iso-orientation slabs tend to connect the centers of ocular dominance bands.

[2] All quantities regarding experimental data are averages over four animals, nm1-nm4, unless stated otherwise. Error margins indicate standard deviations.

[3]Data from area 17 of the cat indicate that in this species, although both elements are present, singularities are more important [23]

[4]The exact form of the algorithm is not essential, however. Algorithms based on similar principles, e.g. the elastic net algorithm [6], predict similar patterns.

## References

[1] Blasdel G.G. and Salama G. (1986), Nature **321**, 579-585.
[2] Blasdel G.G. (1992), J. Neurosci. in press.
[3] Blasdel G.G. (1992), J. Neurosci. in press.
[4] Kohonen T. (1987), Self-Organization and Associative Memory, Springer-Verlag, New York.
[5] Ritter H. and Schulten K. (1988), Biol. Cybern. **60**, 59-71.
[6] Durbin R. and Mitchison M. (1990), Nature **343**, 644-647.
[7] Obermayer K. et al. (1990), Proc. Natl. Acad. Sci. USA **87**, 8345-8349.
[8] Hubel D.H. and Wiesel T.N. (1974), J. Comp. Neurol. **158**, 267-294.
[9] Braitenberg V. and Braitenberg C. (1979), Biol. Cybern. **33**, 179-186.
[10] Swindale N.V. (1982), Proc. R. Soc. Lond. B **215**, 211-230.
[11] Baxter W.T. and Dow B.M. (1989), Biol. Cybern. **61**, 171-182.
[12] Rojer A.S. and Schwartz E.L. (1990), Biol. Cybern. **62**, 381-391.
[13] Malsburg C. (1973), Kybernetik **14**, 85-100.
[14] Takeuchi A. and Amari S. (1979), Biol. Cybern. **35**, 63-72.
[15] Kohonen T. (1982a), Biol. Cybern. **43**, 59-69.
[16] Kohonen T. (1982b), Biol. Cybern. **44**, 135-140.
[17] Linsker R. (1986), Proc. Natl. Acad. Sci. USA **83**, 8779-8783.
[18] Soodak R. (1987), Proc. Natl. Acad. Sci. USA **84**, 3936-3940.
[19] Kammen D.M. and Yuille A.R. (1988), Biol. Cybern. **59**, 23-31.
[20] Miller K.D. et al. (1989), Science **245**, 605-615.
[21] Miller K.D. (1989), Soc. Neurosci. Abs. **15**, 794.
[22] Grinvald A. et al. (1986), Nature **324**, 361-364.
[23] Bonhoeffer T. and Grinvald A. (1991), Nature **353**, 429-431.